# Probabilistic Event Cascades for Alzheimer's disease

**Jonathan Huang**
Stanford University
jhuang11@stanford.edu

**Daniel Alexander**
University College London
d.alexander@cs.ucl.ac.uk

## Abstract

Accurate and detailed models of neurodegenerative disease progression are crucially important for reliable early diagnosis and the determination of effective treatments. We introduce the ALPACA (**Al**zheimer's disease **P**rob**a**bilistic **Ca**scades) model, a generative model linking latent Alzheimer's progression dynamics to observable biomarker data. In contrast with previous works which model disease progression as a fixed event ordering, we explicitly model the variability over such orderings among patients which is more realistic, particularly for highly detailed progression models. We describe efficient learning algorithms for ALPACA and discuss promising experimental results on a real cohort of Alzheimer's patients from the Alzheimer's Disease Neuroimaging Initiative.

## 1 Introduction

Models of disease progression are among the core tools of modern medicine for early disease diagnosis, treatment determination and for explaining symptoms to patients. In neurological diseases, for example, symptoms and pathologies tend to be similar in different diseases. The ordering and severity of those changes, however, provide discrimination amongst different diseases. Thus progression models are key to early differential diagnosis and thus to drug development (for finding the right participants in trials) and for eventual deployment of effective treatments. Despite their utility, traditional models of disease progression [3, 17] have largely been limited to coarse symptomatic staging which divides patients into a small number of groups by thresholding a crude clinical score of how far the disease has progressed. The models are thus only as precise as these crude clinical scores — although providing insight into disease mechanisms, they provide little benefit for early diagnosis or accurate patient staging. With the growing availability of larger datasets consisting of measurements from clinical, imaging and pathological sources, however, more detailed characterizations of disease progression are now becoming feasible and a key hope in medical science is that such models will provide earlier, more accurate diagnosis, leading to more effective development and deployment of emerging treatments. The recent availability of cross sectional datasets such as the Alzheimer's Disease Neuroimaging Initiative data has generated intense speculation in the neurology community about the nature of the cascade of events in AD and the ordering in which biomarkers show abnormality. Several hypothetical models [12, 5, 1] broadly agree, but differ in some ways. Despite early attempts on limited data sets [13], a data driven confirmation of those models remains a pressing need.

Beckett [2] was the first, nearly two decades ago, to propose a data driven model of disease progression using a distribution over orderings of clinical events. This earlier work of [2] considered the progressive loss of physical abilities in ageing persons such as the ability to do heavy work around the house, or to climb up stairs. More recently, Fonteijn et al. [8] developed *event-based models of disease progression* by analyzing ordered series of much finer grained clinical and atrophy events with applications to the study of familial Alzheimer's disease and Huntington's disease, both of which are well-studied autosomal-dominantly inherited neurodegenerative diseases. Examples of events in the model of [8] include (but are not limited to) clinical events (such as a transition from Presymptomatic Alzheimer's to Mild Cognitive Impairment) and the onset of atrophy (reduction of tissue volume). By assuming a single universal ordering of events within the disease progression, the method of [8] is able to scale to much larger collections of events, thus achieving much more detailed characterizations of disease progression compared to that of [2].

The assumption made in [8] of a universal ordering common to all patients within a disease cohort, is a major oversimplification of reality, however, where the event ordering can vary considerably among patients even if it is consistent enough to distinguish different diseases. In practice, the assumption of a universal ordering within the model means we cannot recover the diversity of orderings over population groups and can make fitting the model to patient data unstable. To address the universal ordering problem, our work revisits the original philosophy of [2] by explicitly modeling a distribution over permutations. By carefully considering computational complexity and exploiting modern machine learning techniques, however, we are able to overcome many of its original limitations. For example, where [2] did not model measurement noise, our method can handle a wide range of measurement models. Additionally, like [8], our method can achieve the scalability that is required to produced fine-grained disease progression models. The following is a summary of our main contributions.

- We propose the **Al**zheimer's disease **P**robabilistic **Ca**scades (ALPACA) model, a probabilistic model of disease cascades, allowing for patients to have distinct event orderings.
- We develop efficient probabilistic inference and learning algorithms for ALPACA, including a novel patient "staging" method, which predicts a patient's full trajectory through clinical and atrophy events from sparse and noisy measurement data.
- We provide empirical validation of our algorithms on synthetic data in a variety of settings as well as promising preliminary results for a real cohort of Alzheimer's patients.

## 2 Preliminaries: Snapshots of neurodegenerative disease cascades

We model a neurodegenerative disease cascade as an ordering of a discrete set of $N$ events, $\{e_1, \ldots, e_N\}$. These events represent changes in patient state, such as a sufficiently low score on a memory test for a clinical diagnosis of AD, or the first measurement of tissue pathology, such as significant atrophy in the hippocampus (memory related brain area). An ordering over events is represented as a permutation $\sigma$ which corresponds events to the positions within the ordering at which they occur. We write $\sigma$ as $\sigma(1)|\sigma(2)|\ldots|\sigma(N)$, where $\sigma(j) = e_i$ means that "Event $i$ occurs in position $j$ with respect to $\sigma$". In practice, the ordering $\sigma$ for a particular patient can only be observed indirectly via *snapshots* which probe at a particular point in time whether each event has occurred or not. We denote a snapshot by a vector of $N$ measurements $z = (z_{e_1}, \ldots, z_{e_n})$, where each $z_{e_i}$ is a real valued measurement reflecting a noisy diagnosis as to whether event $i$ of the disease progression has occurred prior to measuring $z$.[1] Were it not for noise within the measurement process, a single snapshot $z$ would partition the event set into two disjoint subsets: events that have occurred already (e.g., $\{e_{\sigma(1)}, \ldots, e_{\sigma(r)}\}$), and events which have yet to occur (e.g., $\{e_{\sigma(r+1)}, \ldots, e_{\sigma(N)}\}$).

Where prior models [8] considered data in which a patient is only associated with a single snapshot (taken at a single time point), we allow for multiple snapshots of a patient to be taken spaced throughout that patient's disease cascade. In this more general case of $k$ snapshots, the event set is partitioned into $k + 1$ disjoint subsets (in the absence of noise). For example, if $\sigma = e_3|e_1|e_4|e_5|e_6|e_2$, then $k = 2$ snapshots might partition the event ordering into sets $X_1 = \{e_1, e_3\}$, $X_2 = \{e_4, e_5\}$, $X_3 = \{e_2, e_6\}$, reflecting that events in $X_1$ occur before events in $X_2$, which occur before events in $X_3$. Such partitions can also be thought of as *partial rankings* over the events (and indeed, we will exploit recent methods for learning with partial rankings in our own approach, [11]). To denote partial rankings, we again use vertical bars, separating the events that occur between snapshots. In the above example, we would write $e_1, e_3|e_4, e_5|e_2, e_6$. This connection between snapshots and partial rankings plays a key role in our inference algorithms in Section 4.1.

Instead of reasoning with continuous snapshot times, we use the fact that *many distinct snapshot times can result in the same partial ranking*, to reason instead with discrete snapshot sets. By *snapshot set*, we refer to the collection of positions in the full event ordering just before each snapshot is taken. In our running example, the snapshot set is $\tau = \{2, 4\}$. Given a full ordering $\sigma$, the partial ranking which arises from snapshot data (assuming no noise) is fully determined by $\tau$. We denote this resulting partial ranking by $\sigma|_\tau$. Thus in our running example, $\sigma|_{\tau=\{2,4\}} = e_1, e_3|e_4, e_5|e_2, e_6$.

## 3 ALPACA: the Alzheimer's disease Probabilistic Cascades model

We now present *ALPACA*, a generative model of noisy snapshots in which the event ordering for each patient is a latent variable. ALPACA makes two main assumptions: (1), that the measured outcomes for each patient are independent of each other and (2), conditioned on the event ordering of each

patient and the time at which a snapshot is taken, the measurements for each event are independent. In contrast with [8], we do *not* assume that multiple snapshot vectors for the same patient are independent of each other. The simplest form of ALPACA is as follows. For each patient $j = 1, \ldots, M$:

1. Draw an ordering of the events $\sigma^{(j)}$ from a *Mallows distribution* $P(\boldsymbol{\sigma}; \sigma_0, \lambda)$ over orderings.
2. Draw a snapshot set $\tau^{(j)}$ from a *uniform* distribution $P(\boldsymbol{\tau})$ over subsets of the event set.
3. For each element of the snapshot set, $\tau_i^{(j)} = \tau_1^{(j)}, \ldots, \tau_{K^{(j)}}^{(j)}$ and for each event $e = e_1, \ldots, e_N$:

    (a) If $\sigma^{-1}(e) \leq \tau_i^{(j)}$ (*i.e., if event $e$ has occurred prior to time $\tau_i^{(j)}$*), draw $z_{i,e}^{(j)} \sim \mathcal{N}(\mu_e^{occurred}, c_e^{occurred})$. Otherwise draw $z_{i,e}^{(j)} \sim \mathcal{N}(\mu_e^{healthy}, c_e^{healthy})$.

In the above basic model, each entry of a snapshot vector, $z_{i,e}^{(j)}$, is generated by sampling from a univariate measurement model (assumed in this case to be Gaussian). If event $e$ has already occurred, the observation $z_{i,e}^{(j)}$ is sampled from the distribution $\mathcal{N}(\mu_e^{occurred}, c_e^{occurred})$ — otherwise $z_{i,e}^{(j)}$ is sampled from a measurement distribution estimated from a control population of healthy individuals, $\mathcal{N}(\mu_e^{healthy}, c_e^{healthy})$. For notational simplicity, we denote the collection of snapshots for patient $j$ by $z_{\cdot,\cdot}^{(j)} = \{z_{i,e}^{(j)}\}_{i=1,\ldots,K^{(j)}, e=1,\ldots,N}$. We remark that the success of our approach does not hinge on the assumption of normality and our algorithms can deal with a variety of measurement models. For example, certain clinical events (such as the loss of the ability to pass a memory test) are more naturally modeled as discrete observations and can trivially be incorporated into the current model.

The prior distribution over possible event orderings is assumed to take the form of the well known *Mallows distribution*, which has been used in a number of other application areas such as NLP, social choice, and psychometrics ([6, 15, 18]), and has the following probability mass function over orderings: $P(\boldsymbol{\sigma} = \sigma; \sigma_0, \lambda) \propto \exp\left(-\lambda d_K(\sigma, \sigma_0)\right)$, where $d_K(\cdot, \cdot)$ is the *Kendall's tau* distance metric on orderings. The Kendall's tau distance penalizes the number of *inversions*, or pairs of events for which $\sigma$ and $\sigma_0$ disagree over relative ordering. Mallows models are analogous to normal distributions in that $\sigma_0$ can be interpreted as the *mean* or *central ordering* and $\lambda$ as a measure of the "spread" of the distribution. Both parameters are viewed as fixed quantities to be estimated via the empirical Bayesian approach outlined in Section 4.

The choices of the Mallows model for orderings and the uniform distribution for snapshot sets are particularly convenient for clinical settings in which the number of subjects may be limited, since the small number of parameters of the model (which scales linearly in $N$) sufficiently constrains learning, and eases our discussion of inference and learning in Section 4. However, as we discuss in Section 5, the parametric assumptions made in the most basic form of ALPACA can be considerably relaxed without impacting the computational complexity of learning. Our algorithms are thus applicable for more general classes of distributions over orderings as well as snapshot sets.

**Application to patient staging.** With respect to the event-based characterization of disease progression, a critical problem is that of *patient staging*, the problem of determining the extent to which a disease has progressed for a particular patient given corresponding measurement data. ALPACA offers a simple and natural formulation of the patient staging problem as a probabilistic inference query. In particular, given the measurements corresponding to a particular patient, we perform patient staging by: (1) computing a posterior distribution over the event ordering $\boldsymbol{\sigma}^{(j)}$, then (2) computing a posterior distribution over the most recent element of the snapshot set $\boldsymbol{\tau}^{(j)}$.

To visualize the posterior distribution over the event ordering $\boldsymbol{\sigma}^{(j)}$, we plot a simple "first-order staging diagram", displaying the probability that event $e$ has occurred (or will occur) in position $q$ according to the posterior. Two major features differentiate ALPACA from traditional patient staging approaches, in which patients are binned into a small number of imprecisely defined stages. In particular, our method is more fine-grained, allowing for a detailed picture of what the patient has undergone as well as a prediction of what is to come next. Moreover, ALPACA has well-defined probabilistic semantics, allowing for a rigorous probabilistic characterization of uncertainty.

## 4 Inference algorithms and parameter estimation

In this section we describe tractable inference and parameter estimation procedures for ALPACA.

### 4.1 Inference.

Given a collection of $K^{(j)}$ snapshots for a patient $j$, the critical inference problem that we must solve is that of computing a posterior distribution over the latent event order and snapshot set for that patient. Despite the fact that all latent variables are discrete, however, computing this

posterior distribution can be nontrivial due to the super-exponential size of the state space (which is $O(N! \times \binom{N}{K^{(j)}})$), for which there exist no tractable exact inference algorithms.

We thus turn to a Gibbs sampling approximation. Directly applying the Gibbs sampler to the model is difficult however. One reason is that it is not obvious how to tractably sample the event ordering $\boldsymbol{\sigma}$ conditioned on its Markov blanket, given that the corresponding likelihood function is not conjugate prior to the Mallows model. Instead, noting that the snapshots depend on $(\boldsymbol{\sigma}, \boldsymbol{\tau})$ only through the partial ranking $\boldsymbol{\gamma} \equiv \boldsymbol{\sigma}|_{\boldsymbol{\tau}}$, our Gibbs sampler operates on an augmented model in which the partial ranking $\boldsymbol{\gamma}$ is first generated (deterministically) from $\boldsymbol{\sigma}$ and $\boldsymbol{\tau}$, and the snapshots are then generated conditioned on the partial ranking $\boldsymbol{\gamma}$. See Fig. 1(a) for a Bayes net representation. This augmented model is equivalent to the original model, but has the advantage that it reduces the sampling step for the event ordering $\boldsymbol{\sigma}$ to a well understood problem (described below). Our sampler thus alternates between sampling $\boldsymbol{\sigma}$ and jointly sampling $(\boldsymbol{\gamma}, \boldsymbol{\tau})$ from the following conditional distributions:
$$\sigma^{(j)} \sim P(\boldsymbol{\sigma} \mid \boldsymbol{\gamma} = \gamma^{(j)}, \boldsymbol{\tau} = \tau^{(j)} \; ; \; \sigma_0, \phi), \qquad (\gamma^{(j)}, \tau^{(j)}) \sim P(\boldsymbol{\gamma}, \boldsymbol{\tau} \mid \boldsymbol{\sigma} = \sigma^{(j)} \; , \; z_{:,:}^{(j)}). \qquad (4.1)$$
Observe that since the snapshot set $\boldsymbol{\tau}$ is fully determined by the partial ranking $\boldsymbol{\gamma}$, it is not necessary to condition on $\boldsymbol{\tau}$ in Equation 4.1 (left). Similarly in Equation 4.1 (right), since $\boldsymbol{\gamma}$ is fully determined given both the event ordering $\boldsymbol{\sigma}$ and the snapshot set $\boldsymbol{\tau}$, one can sample $\boldsymbol{\tau}$ first, and deterministically reconstruct $\boldsymbol{\gamma}$. Therefore the Gibbs sampling updates are:
$$\sigma^{(j)} \sim P(\boldsymbol{\sigma} \mid \boldsymbol{\gamma} = \gamma^{(j)} \; ; \; \sigma_0, \phi), \qquad \tau^{(j)} \sim P(\boldsymbol{\tau} \mid \boldsymbol{\sigma} = \sigma^{(j)} \; , \; z_{:,:}^{(j)}). \qquad (4.2)$$
While the Gibbs sampling updates here effectively reduce the inference problem to smaller inference problems, the state spaces over $\boldsymbol{\sigma}$ and $\boldsymbol{\tau}$ still remain intractably large (with cardinalities $O(N!)$ and $O(\binom{N}{K^{(j)}})$, respectively). In the remainder of this section, we show how to exploit even further structure within each of the conditional distributions over $\boldsymbol{\sigma}$ and $\boldsymbol{\tau}$ for efficient inference. As a result, *we are able to carry out Gibbs sampling operations efficiently and exactly*.

**Sampling event orderings.** To sample $\sigma^{(j)}$ from the conditional distribution in Equation 4.2, we must condition a Mallows prior on the partial ranking $\boldsymbol{\gamma} = \gamma^{(j)}$. This precise problem has in fact been discussed in a number of works [4, 14, 15, 9]. In our experiments, we use the method of Huang [9] which explicitly computes a representation of the posterior, from which one can efficiently (and exactly) draw independent samples.

**Sampling snapshot sets.** We now turn to the problem of sampling a snapshot set $\tau^{(j)}$ of size $K^{(j)}$ from Equation 4.2 (right). Note first that if $K^{(j)}$ is small (say, less than 3), then one can exhaustively compute the posterior probability of each of the $\binom{N}{K^{(j)}}$ $K^{(j)}$-subsets and draw a sample from a tabular representation of the posterior. For larger $K^{(j)}$, however, the exhaustive approach is intractable. In the following, we present a dynamic programming algorithm for sampling snapshot sets with running time much lower than the exhaustive setting (even for small $K^{(j)}$). Our core insight is to exploit conditional independence relations within the posterior distribution over snapshot sets. That such independence relations exist may not seem surprising due to the simplicity of the uniform prior over snapshot sets — but on the other hand, note that the individual times of a snapshot set drawn from the uniform distribution over $K^{(j)}$-subsets are *not* a priori independent of each other (they could not be, as the total number of times is observed and fixed to be $K^{(j)}$). As we show in the following, however, we can bijectively associate each snapshot set with a trajectory through a certain grid. With respect to this grid-based representation of snapshot sets, we then show that the posterior distribution can be viewed as that of a particular hidden Markov model (HMM).

We will consider the set $\mathcal{G} = \{(x, y) : 0 \leq x \leq K^{(j)} \text{ and } 0 \leq y \leq N - K^{(j)}\}$. $\mathcal{G}$ is a grid (depicted in Fig. 1(b)) which we will visualize with $(K^{(j)}, N - K^{(j)})$ in the upper left corner and $(0, 0)$ in the lower right corner. Let $\mathcal{P}_{\mathcal{G}}$ denote the collection of *staircase walks* (paths which never go up or to the left) through the grid $\mathcal{G}$ starting and ending at the corners $(K^{(j)}, N - K^{(j)})$ and $(0, 0)$, respectively. An example staircase walk is outlined in blue in Figure 1(b). It is not difficult to verify that every element in $\mathcal{P}_{\mathcal{G}}$ has length $N$ (i.e., every staircase walk traverses exactly $N$ edges in the grid).

Given a grid $\mathcal{G}$, we can now state a one-to-one correspondence between the staircase walks in $\mathcal{P}_{\mathcal{G}}$ with the $K^{(j)}$-subsets of $N$. To establish the correspondence, we first associate each edge of the grid to the sum of the indices of the starting node of that edge. Hence the edge from $(x_1, y_1)$ to $(x_2, y_2)$ is associated with the number $x_1 + y_1$. Given any staircase walk $p = ((x_0, y_0), (x_1, y_1), ..., (x_N, y_N))$ in $\mathcal{P}_{\mathcal{G}}$, we associate $p$ to the subset of events in $\{1, \ldots, N\}$ corresponding to the subset of edges of $p$ which point downwards. It is not difficult to show that this association is in fact, bijective (i.e., given a snapshot set $\tau$, there is a unique staircase walk $p_{\tau}$ mapping to $\tau$).

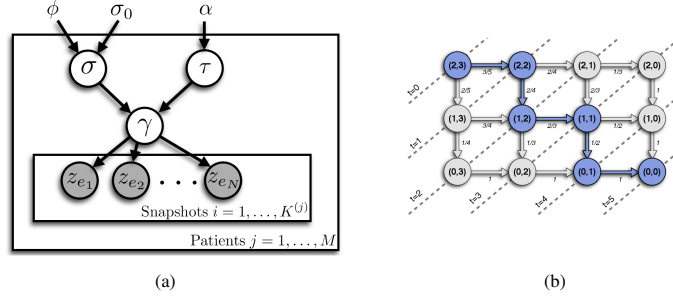

(a)                                                                    (b)

Figure 1: (a): Bayesian network representation of our model (augmented by adding the partial ranking $\gamma$). (b): Grid structured state space $\mathcal{G}$ for sampling snapshot sets with edges labeled with transition probabilities according to Equation 4.3. In this example, $N = 5$ and $K^{(j)} = 2$. The example path (highlighted) is $p = ((2,3), (2,2), (1,2), (1,1), (0,1), (0,0))$, corresponding to the snapshot set $\tau = \{4, 2\}$.

We now show that our encoding of $K^{(j)}$-subsets as staircase walks allows for the posterior over $\boldsymbol{\tau}$ in Equation 4.2 to factor with respect to a hidden Markov model. Conditioned on $\boldsymbol{\sigma} = \sigma^{(j)}$, we define an HMM over $\mathcal{G}$ with the following transition and observation probabilities, respectively:

$$P((\boldsymbol{x_t}, \boldsymbol{y_t}) \mid (\boldsymbol{x_{t-1}}, \boldsymbol{y_{t-1}}) = (x, y)) \equiv \begin{cases} \frac{x}{x+y} & \text{if } (\boldsymbol{x_t}, \boldsymbol{y_t}) = (x - 1, y) \\ \frac{y}{x+y} & \text{if } (\boldsymbol{x_t}, \boldsymbol{y_t}) = (x, y - 1) \\ 0 & \text{otherwise} \end{cases}, \tag{4.3}$$

$$L(z^{(j)}_{\cdot, \sigma(N-t)} \mid (\boldsymbol{x_t}, \boldsymbol{y_t}) = (x_t, y_t)) \equiv \phi(x_t, x_t + y_t; z^{(j)}_{1, \sigma(N-t)}, \ldots, z^{(j)}_{K^{(j)}, \sigma(N-t)}), \tag{4.4}$$

where $\phi(v, e; z_1, \ldots, z_K) \equiv \prod_{i=1}^{v-1} P(z_i; \mu^{healthy}_e, c^{healthy}_e) \prod_{i=v}^{K} P(z_i; \mu^{occurred}_e, c^{occurred}_e)$.

The initial state is set to $(\boldsymbol{x_0}, \boldsymbol{y_0}) = (K^{(j)}, N - K^{(j)})$ and the chain terminates when $(\boldsymbol{x}, \boldsymbol{y}) = (0, 0)$. Note that sample trajectories from the above HMM are staircase walks with probability one.

**Proposition 1.** *Conditioned on $\boldsymbol{\sigma} = \sigma^{(j)}$, the posterior probability $P(\boldsymbol{\tau} = \tau^{(j)} \mid \boldsymbol{\sigma} = \sigma^{(j)}, z^{(j)}_{\cdot, \cdot})$ is equal to the posterior probability of the staircase walk $p_{\tau^{(j)}}$ under the hidden Markov model defined by Equations 4.3 and 4.4.*

To sample a snapshot set from the conditional distribution in Equation 4.2, we therefore sample staircase walks from the above HMM and convert the resulting samples to snapshot sets.

**Time Complexity of a single Gibbs iteration.** We now consider the computational complexity of our inference procedures. First observe that the complexity of sampling from the posterior distribution of a Mallows model conditioned on a partial ranking is $O(N^2)$ [9]. We claim that the complexity of sampling a snapshot set is also $O(N^2)$. To see why, note that the complexity of the Backwards algorithm for HMMs is squared in the number of states and linear in the number of time steps. In our case, the number of states is $K^{(j)}(N - K^{(j)})$ and the number of time steps is $N$. Thus in the worst case (where $K^{(j)} \approx N/2$), the complexity of naively sampling a staircase walk is $O(N^5)$. However we can exploit additional problem structure. First, since the HMM transition matrix is sparse (each state transitions to at most two states), the Backwards algorithm can be performed in $O(N \cdot \#(states))$ time. Second, since the grid coordinates corresponding to the current state at time $T$ are constrained to sum to $N - T$, the size of the *effective* state space is reduced to $O(N)$ rather than $O(K^{(j)}(N - K^{(j)}))$. Thus in the worst case, the running time complexity can in turn be reduced to $O(N^2)$ and even linear time $O(N)$ when $K^{(j)} \sim O(1)$. In conclusion, the total complexity of a single Gibbs iteration requires at most $O(N^2)$ operations.

**Mixing considerations.** Under mild assumptions, it is not difficult to establish ergodicity of our Gibbs sampler, showing that the sampling distribution must eventually converge to the desired posterior. The one exception is when the size of the snapshot set is one less than the number of events ($K^{(j)} = N - 1$). In this exceptional case,[2] the grid $\mathcal{G}$ has size $N \times 1$, forcing the Gibbs sampler to be deterministic. As a result, the Markov chain defined by the Gibbs sampler is not irreducible and hence not ergodic. We have:

**Proposition 2.** *The Gibbs sampler is ergodic on its state space if and only if $K^{(j)} < N - 1$.*

Even when $K^{(j)} < N - 1$, mixing times for the chain can be longer for larger snapshot sets (where $K^{(j)}$ is close to $N - 1$). For example, when $K^{(j)} = N - 2$, it is possible to show that the $T^{th}$ ordering in the Gibbs chain can differ from the $(T + 1)^{th}$ ordering by at most an adjacent swap. Consequently, since it requires $O(N^2)$ adjacent swaps (in the worst case) to reach the mode of the posterior distribution with nonzero probability, we can lower bound the mixing time in this case by $O(N^2)$ steps. For smaller $K^{(j)}$, the Gibbs sampler is able to make larger jumps in state space and indeed, for these chains, we observe faster mixing times in practice.

## 4.2 Parameter estimation.

Given a snapshot dataset $\{z^{(j)}\}_{j=1,\ldots,M}$, we now discuss how to estimate the ALPACA model parameters $(\sigma_0, \lambda)$ by maximizing the marginal log likelihood: $\ell(\sigma_0, \lambda) = \sum_{j=1}^{M} \log P(z^{(j)}|\sigma_0, \lambda)$. Currently we obtain point estimates of model parameters, but fuller Bayesian approaches are also possible. Our approach uses Monte Carlo expectation maximization (EM) to alternate between the following two steps given an initial setting of model parameters $(\sigma_0^{(0)}, \lambda^{(0)})$.

**E-step.** For each patient in the cohort, use the inference algorithm described in Section 4.1 to obtain a draw from the posterior distribution $P(\boldsymbol{\sigma}^{(j)}, \boldsymbol{\tau}^{(j)}|z^{(j)}, \sigma_0, \lambda)$. Note that multiple draws can also be taken to reduce the variance of the E-step.

**M-step.** Given the draws obtained via the E-step, we can now apply standard Mallows model estimation algorithms (see [7, 16, 15]) to optimize for the parameters $\sigma_0$ and $\lambda$ given the sampled ordering for each patient. Optimizing for $\lambda$, for example, is a one-dimensional convex optimization [16]. Optimizing for $\sigma_0$ (sometimes called the *consensus ranking problem*) is known to be NP-hard. Our implementation uses the Fligner and Verducci heuristic [7] (which is known to be an unbiased estimator of $\sigma_0$) followed by local search, but more sophisticated estimators exist [16]. Note that the sampled snapshot sets ($\{\tau^{(j)}\}$) do not play a role in the M-step described here, but can be used to estimate parameters for the more complex snapshot set distributions described in Section 5.

**Complexity of EM.** The running time of a single iteration of our E-step requires $O(N^2 T_{Gibbs} M)$ time, where $T_{Gibbs}$ is the number of Gibbs iterations. The running time of the M-step is $O(N^2 M)$ (assuming a single sample per patient), and is therefore dominated by the E-step complexity.

# 5 Extensions of the basic model

**Generalized ordering models.** The classical Mallows model for orderings is often too limited for real datasets in its lack of flexibility. One limitation is that the positional variances of all of the events are governed by just a single parameter, $\lambda$. In clinical datasets, it is more conceivable that different biomarkers within a disease cascade change over different timescales, thus leading to higher positional variance for certain events and lower positional variance for others.

Fortunately our approach applies to any class of distributions for which one can efficiently condition on partial ranking observations. In our experiments (Section 6), we achieve more flexibility using the *generalized Mallows model* [7, 16], which includes the classical Mallows model as a special case and allows for the positional variance of each event $e$ to be governed by its own corresponding parameter $\lambda_e$. Generalized Mallows models are in turn a special case of the recently introduced *hierarchical riffle independent models* [10] which allow one to capture dependencies among small subsets of events. Huang et al. ([11]), in particular, proved that these hierarchical riffle independent models form a natural conjugate prior family for partial ranking likelihood functions and introduced efficient algorithms for conditioning on partial ranking observations.

It is finally interesting to note that it would *not* be trivial to use traditional Markov chains to capture the dependencies in the event sequence due to the fact that observations come in snapshot form instead of being indexed by time as they would be in an ordinary hidden Markov model. Thus in order to properly perform inference, one would have to infer an HMM posterior with respect to each of the permutations of the event set, which is computationally harder.

**Generalized snapshot set models.** Going beyond the uniform distribution, ALPACA can also efficiently handle a more general class of snapshot set distribution by observing that any distribution parametrizable as a Markov chain over the grid $\mathcal{G}$ that generates staircase walks can be substituted for the uniform distribution with exactly the same time complexity of Gibbs sampling. As a cautionary remark, we note that allowing for these more general models without additional constraints can sometimes lead to instabilities in parameter estimation. A simple constrained Markov chain that we have successfully used in experiments parameterizes transition probabilities such that a staircase

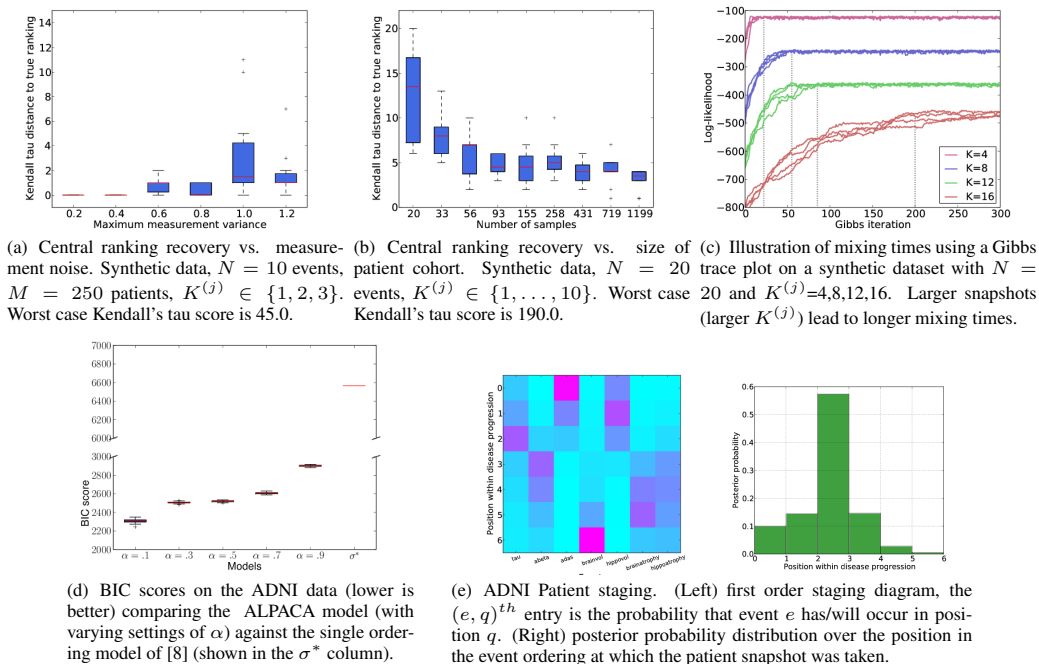

(a) Central ranking recovery vs. measurement noise. Synthetic data, $N = 10$ events, $M = 250$ patients, $K^{(j)} \in \{1, 2, 3\}$. Worst case Kendall's tau score is 45.0.

(b) Central ranking recovery vs. size of patient cohort. Synthetic data, $N = 20$ events, $K^{(j)} \in \{1, \dots, 10\}$. Worst case Kendall's tau score is 190.0.

(c) Illustration of mixing times using a Gibbs trace plot on a synthetic dataset with $N = 20$ and $K^{(j)}$=4,8,12,16. Larger snapshots (larger $K^{(j)}$) lead to longer mixing times.

(d) BIC scores on the ADNI data (lower is better) comparing the ALPACA model (with varying settings of $\alpha$) against the single ordering model of [8] (shown in the $\sigma^*$ column).

(e) ADNI Patient staging. (Left) first order staging diagram, the $(e, q)^{th}$ entry is the probability that event $e$ has/will occur in position $q$. (Right) posterior probability distribution over the position in the event ordering at which the patient snapshot was taken.

Figure 2: Experimental results

walk moves down at node $(x, y)$ in the grid $\mathcal{G}$ with probability proportional to $\alpha x$ and to the left with probability proportional to $(1 - \alpha)y$. Setting $\alpha = 1/2$ recovers the uniform distribution. Setting $0 \le \alpha < 1/2$, however, reflects a prior bias for snapshots to have been taken earlier in the disease cascade, while setting $1/2 < \alpha \le 1$ reflects a prior bias for snapshots to have been taken later in the disease cascade. Thus $\alpha$ intuitively allows us to interpolate between early and late detection.

## 6   Experiments

**Synthetic data experiments**   We first validate ALPACA on synthetic data. Since we are interested in the ability of the model to recover the true central ranking, we evaluate based on the Kendall's tau distance between the ground truth central ranking and the central rankings learned by our algorithms.

To understand how learning is impacted by measurement noise, we simulate data from models in which the means $\mu^{healthy}$ and $\mu^{occurred}$ are fixed to be 0 and 1, respectively and variances are selected uniformly at random from the interval $(0, c_e^{MAX})$, then learn model parameters from the simulated data. Fig. 2(a) illustrates the results on a problem with $N = 10$ events and 250 patients (with $K^{(j)}$ set to be 1, 2, or 3 randomly for each patient) as $c_e^{MAX}$ varies between $[0.2, 1.2]$. As shown in the figure, we obtain nearly perfect performance for low measurement noise with recovery rates degrading gracefully with higher measurement noise levels.

We also show results on a larger problem with $N = 20$ events, $c_e = 0.1$, and $K^{(j)}$ drawn uniformly at random from $\{1, \dots, 10\}$. Varying the cohort size, this time, Fig. 2(b) shows, as expected, that recovery rates for the central ordering improve as the number of patients increases. Note that with 20 events, it would be utterly intractable to use brute force inference algorithms, but our algorithms can process a patient's measurements in roughly 3 seconds on a laptop.

In both experiments for Figs. 2(a) and 2(b), we discard the first 200 burn-in iterations of Gibbs, but it is often sufficient to discard much fewer iterations. To illustrate mixing behavior, Fig. 2(c) shows example Gibbs trace plots with $N = 20$ events and varying sizes of the snapshot set, $K^{(j)}$. We observe that mixing time increases as $K^{(j)}$ increases, confirming the discussion of mixing (Sec. 4.1).

**The ADNI dataset.**   We also present a preliminary analysis of a cohort with a total number of 347 subjects (including 83 control subjects) from the Alzheimer's Disease Neuroimaging Institute (ADNI). We derive seven typical biomarkers associated with the onset of Alzheimers:  (1) the total tau level in cerebral spinal fluid (CSF) [**tau**], (2) the total A$\beta$42 level in CSF [**abeta**], (3) the total ADAS cognitive assessment score [**adas**], (4) brain volume [**brainvol**], (5) hippocampal volume [**hippovol**], (6) brain atrophy rate [**brainatrophy**], and (7) hippocampal atrophy rate

[**hippoatrophy**]. Due to the small number of measured events in the ADNI data, it is possible to apply the model of Fonteijn et al. [8] (which assumes that all patients follow a single ordering $\sigma^*$) by searching exhaustively over the collection of all $7! = 5040$ orderings. We compare the ALPACA model against the single ordering model via BIC scores (shown in Fig. 2(d)). We fit our model five times, with the bias parameter $\alpha$ (described in Section 5) set to $.1, .3, .5, .7, .9$. We use a single Gaussian for each of the *healthy* and *occurred* measurement distributions (as described in [8]), assuming that all patients in the control group are healthy.[3]

The results show that by allowing for the event ordering $\sigma$ to vary across patients, the AL-PACA model significantly outperforms the single ordering model (shown in the $\sigma^*$ column) in BIC score with respect to all of the tried settings of $\alpha$. Further, we observe that setting $\alpha = 0.1$ minimizes the BIC, reflecting the fact, we conjecture, that many of the patients in the ADNI cohort are in the earlier stages of Alzheimers. The optimal central ordering inferred by the Fonteijn model is: $\sigma^* = $ **adas|hippovol|hippoatrophy|brainatrophy|abeta|tau|brainvol**, while ALPACA infers the central ordering: $\sigma_0 = $ **adas|hippovol|abeta|hippoatrophy|tau|brainatrophy|brainvol**. Observe that the two event orderings are largely in agreement with each other with CSF A$\beta$42 and CSF tau events shifted to being earlier in the event ordering, which is more consistent with current thinking in neurology [12, 5, 1], which places the two CSF events first. Note that **adas** is first in both orderings as it was used to classify the patients — thus its position is somewhat artificial. It is surprising that the hippocampal volume and atrophy events are inferred in both models to occur before the CSF events [13], but we believe that this may be due to the significant proportion of misdiagnosed patients in the data. These misdiagnosed patients still have heavy atrophy in the hippocampus, which is a common pathology among many neurological conditions (other dementias and psychiatric disorders), but a change in CSF A$\beta$ is much more specific to AD. Future work will adapt the model for robustness to these misdiagnoses and other outliers.

Finally, Fig. 2(e) shows the patient staging result for an example patient from the ADNI data. The left matrix visualizes the probability that each event will occur in each position of the event ordering given snapshot data from this patient, while the right histogram visualizes where in the event ordering the patient was situated when the snapshot was taken.

## 7    Conclusions

We have developed the **Al**zheimer's disease **P**rob**a**bilistic **Ca**scades model for event ordering within the Alzheimer's disease cascade. In its most basic form, ALPACA is a simple model with generative semantics, allowing one to learn the central ordering of events that occur within a disease progression as well as to quantify the variance of this ordering across patients. Our preliminary results show that relaxing the notion that a single ordering over events exists for all patients allows ALPACA to achieve a much better fit to snapshot data from a cohort of Alzheimer's patients.

One of our main contributions is to show how the combinatorial structure of event ordering models can be exploited for algorithmic efficiency. While exact inference remains intractable for ALPACA, we have presented a simple MCMC based procedure which uses dynamic programming as a subroutine for highly efficient inference.

There may exist biomarkers for Alzheimer's which are more effective than those considered in our current work for the purposes of patient staging. Identifying such biomarker events remains an open question crucial to the success of data-driven models of disease cascades. Fortunately, one of the main advantages of ALPACA lies in its extensibility and modularity. We have discussed several such possible extensions, from more general measurement models to more general riffle independent ordering models. Additionally, with the ability to scale gracefully with problem size as well as to handle noise, we believe that the ALPACA model will be applicable to many other Alzheimer's datasets as well as datasets for other neurodegenerative diseases.

**Acknowledgements**

J. Huang is supported by a NSF Computing Innovation Fellowship. The EPSRC support D. Alexander's work on this topic with grant EP/J020990/01. The authors also thank Dr. Jonathan Schott, UCL Dementia Centre, and Dr. Jonathan Bartlett, London School of Hygiene and Tropical Medicine, for preparation of the data and help with interpretation of the results.

## Footnotes

[1]For notational simplicity, we assume that measurements corresponding to each event are scalar valued. However, our model extends trivially to more complicated measurements.

[2]Note that to have so many snapshots for a single patient would be rare indeed.

[3]We note that this assumption is a major oversimplification as some of the control subjects are likely affected by some non-AD neurodegenerative disease. Due to these difficulties in obtaining ground truth data, however, estimating accurate measurement models can sometimes be a limitation.

# References

[1] Paul S. Aisen, Ronald C. Petersen, Michael C. Donohue, Anthony Gamst, Rema Raman, Ronald G. Thomas, Sarah Walter, John Q. Trojanowski, Leslie M. Shaw, Laurel A. Beckett, Clifford R. Jack, William Jagust, Arthur W. Toga, Andrew J. Saykin, John C. Morris, Robert C. Green, and Michael W. Weiner. The alzheimer's disease neuroimaging initiative: progress report and future plans. *Alzheimers dementia the journal of the Alzheimers Association*, 6(3):239–246, 2010.

[2] Laurel Beckett. *Maximum likelihood estimation in Mallows's model using partially ranked data.*, pages 92–107. New York: Springer-Verlag, 1993.

[3] H. Braak and E. Braak. Neuropathological staging of alzheimer-related changes. *Acta Neuropathol.*, 82:239–259, 1991.

[4] Ludwig M. Busse, Peter Orbanz, and Joachim Buhmann. Cluster analysis of heterogeneous rank data. In *The 24th Annual International Conference on Machine Learning*, ICML '07, Corvallis, Oregon, June 2007.

[5] A Caroli and G B Frisoni. The dynamics of alzheimer?s disease biomarkers in the alzheimer's disease neuroimaging initiative cohort. *Neurobiology of Aging*, 31(8):1263–1274, 2010.

[6] Harr Chen, S. R. K. Branavan, Regina Barzilay, and David R. Karger. Global models of document structure using latent permutations. In *Proceedings of Human Language Technologies: The 2009 Annual Conference of the North American Chapter of the Association for Computational Linguistics*, NAACL '09, pages 371–379, Stroudsburg, PA, USA, 2009. Association for Computational Linguistics.

[7] Michael Fligner and Joseph Verducci. Mulistage ranking models. *Journal of the American Statistical Association*, 83(403):892–901, 1988.

[8] Hubert M. Fonteijn, Marc Modat, Matthew J. Clarkson, Josephine Barnes, Manja Lehmann, Nicola Z. Hobbs, Rachael I. Scahill, Sarah J. Tabrizi, Sebastien Ourselin, Nick C. Fox, and Daniel C. Alexander. An event-based model for disease progression and its application in familial alzheimer's disease and huntington's disease. *NeuroImage*, 60(3):1880 – 1889, 2012.

[9] Jonathan Huang. *Probabilistic Reasoning and Learning on Permutations: Exploiting Structural Decompositions of the Symmetric Group*. PhD thesis, Carnegie Mellon University, 2011.

[10] Jonathan Huang and Carlos Guestrin. Learning hierarchical riffle independent groupings from rankings. In *International Conference on Machine Learning (ICML 2010)*, Haifa, Israel, June 2010.

[11] Jonathan Huang, Ashish Kapoor, and Carlos Guestrin. Efficient probabilistic inference with partial ranking queries. In *Conference on Uncertainty in Artificial Intelligence*, Barcelona, Spain, July 2011.

[12] Clifford R Jack, David S Knopman, William J Jagust, Leslie M Shaw, Paul S Aisen, Michael W Weiner, Ronald C Petersen, and John Q Trojanowski. Hypothetical model of dynamic biomarkers of the alzheimer's pathological cascade. *The Lancet Neurology 1*, 9:119–128, January 2010.

[13] Clifford R. Jack, Prashanthi Vemuri, Heather J. Wiste, Stephen D. Weigand, Paul S. Aisen, John Q. Trojanowski, Leslie M. Shaw, Matthew A. Bernstein, Ronald C. Petersen, Michael W. Weiner, and David S. Knopman. Evidence for ordering of alzheimer disease biomarkers. *Archives of Neurology*, 2011.

[14] Guy Lebanon and Yi Mao. Non-parametric modeling of partially ranked data. In John C. Platt, Daphne Koller, Yoram Singer, and Sam Roweis, editors, *Advances in Neural Information Processing Systems 20*, NIPS '07, pages 857–864, Cambridge, MA, 2008. MIT Press.

[15] Tyler Lu and Craig Boutilier. Learning mallows models with pairwise preferences. In *The 28th Annual International Conference on Machine Learning*, ICML '11, Bellevue, Washington, June 2011.

[16] Marina Meila, Kapil Phadnis, Arthur Patterson, and Jeff Bilmes. Consensus ranking under the exponential model. Technical Report 515, University of Washington, Statistics Department, April 2007.

[17] Rachael I. Scahill, Jonathan M. Schott, John M. Stevens, Martin N. Rossor, and Nick C. Fox. Mapping the evolution of regional atrophy in alzheimer's disease: Unbiased analysis of fluid-registered serial mri. *Proceedings of the National Academy of Sciences*, 99(7):4703–4707, 2002.

[18] Mark Steyvers, Michael Lee, Brent Miller, and Pernille Hemmer. The wisdom of crowds in the recollection of order information. In Y. Bengio, D. Schuurmans, J. Lafferty, C. K. I. Williams, and A. Culotta, editors, *Advances in Neural Information Processing Systems 22*, pages 1785–1793. 2009.

